# Learning Prototype Models for Tangent Distance

**Trevor Hastie***
Statistics Department
Sequoia Hall
Stanford University
Stanford, CA 94305
email: trevor@playfair.stanford.edu

**Patrice Simard**
AT&T Bell Laboratories
Crawfords Corner Road
Holmdel, NJ 07733
email: patrice@neural.att.com

**Eduard Säckinger**
AT&T Bell Laboratories
Crawfords Corner Road
Holmdel, NJ 07733
email: edi@neural.att.com

## Abstract

Simard, LeCun & Denker (1993) showed that the performance of nearest-neighbor classification schemes for handwritten character recognition can be improved by incorporating invariance to specific transformations in the underlying distance metric — the so called *tangent distance*. The resulting classifier, however, can be prohibitively slow and memory intensive due to the large amount of prototypes that need to be stored and used in the distance comparisons. In this paper we develop rich models for representing large subsets of the prototypes. These models are either used singly per class, or as basic building blocks in conjunction with the K-means clustering algorithm.

## 1  INTRODUCTION

Local algorithms such as K-nearest neighbor (NN) perform well in pattern recognition, even though they often assume the simplest distance on the pattern space. It has recently been shown (Simard et al. 1993) that the performance can be further improved by incorporating invariance to specific transformations in the underlying distance metric — the so called *tangent distance*. The resulting classifier, however, can be prohibitively slow and memory intensive due to the large amount of prototypes that need to be stored and used in the distance comparisons.

In this paper we address this problem for the tangent distance algorithm, by developing rich models for representing large subsets of the prototypes. Our leading example of prototype model is a low-dimensional (12) hyperplane defined by a point and a set of basis or tangent vectors. The components of these models are learned from the training set, chosen to minimize the average *tangent distance* from a subset of the training images — as such they are similar in flavor to the Singular Value Decomposition (SVD), which finds closest hyperplanes in Euclidean distance. These models are either used singly per class, or as basic building blocks in conjunction with K-means and LVQ. Our results show that not only are the models effective, but they also have meaningful interpretations. In handwritten character recognition, for instance, the main tangent vector learned for the the digit "2" corresponds to addition/removal of the loop at the bottom left corner of the digit; for the 9 the fatness of the circle. We can therefore think of some of these learned tangent vectors as representing additional invariances derived from the training digits themselves. Each learned prototype model therefore represents very compactly a large number of prototypes of the training set.

## 2  OVERVIEW OF TANGENT DISTANCE

When we look at handwritten characters, we are easily able to allow for simple transformations such as rotations, small scalings, location shifts, and character thickness when identifying the character. Any reasonable automatic scheme should similarly be insensitive to such changes.

Simard et al. (1993) finessed this problem by generating a parametrized 7-dimensional manifold for each image, where each parameter accounts for one such invariance. Consider a single invariance dimension: rotation. If we were to rotate the image by an angle $\theta$ prior to digitization, we would see roughly the same picture, just slightly rotated. Our images are $16 \times 16$ grey-scale pixelmaps, which can be thought of as points in a 256-dimensional Euclidean space. The rotation operation traces out a smooth one-dimensional curve $X_i(\theta)$ with $X_i(0) = X_i$, the image itself. Instead of measuring the distance between two images as $D(X_i, X_j) = \|X_i - X_j\|$ (for any norm $\|\cdot\|$), the idea is to use instead the rotation-invariant $D^I(X_i, X_j) = \min_{\theta_i, \theta_j} \|X_i(\theta_i) - X_j(\theta_j)\|$. Simard et al. (1993) used 7 dimensions of invariance, accounting for horizontal and vertical location and scale, rotation and shear and character thickness.

Computing the manifold exactly is impossible, given a digitized image, and would be impractical anyway. They approximated the manifold instead by its tangent

plane at the image itself, leading to the tangent model $\tilde{X}_i(\theta) = X_i + T_i\theta$, and the *tangent distance* $D^T(X_i, X_j) = \min_{\theta_i, \theta_j} \left\| \tilde{X}_i(\theta_i) - \tilde{X}_j(\theta_j) \right\|$. Here we use $\theta$ for the 7-dimensional parameter, and for convenience drop the tilde. The approximation is valid locally, and thus permits local transformations. Non-local transformations are not interesting anyway (we don't want to flip 6s into 9s; shrink all digits down to nothing.) See Säckinger (1992) for further details. If $\|\cdot\|$ is the Euclidean norm, computing the tangent distance is a simple least-squares problem, with solution the square-root of the residual sum-of-squares of the residuals in the regression with response $X_i - X_j$ and predictors $(-T_i : T_j)$.

Simard et al. (1993) used $D^T$ to drive a 1-NN classification rule, and achieved the best rates so far—2.6%—on the official test set (2007 examples) of the USPS data base. Unfortunately, 1-NN is expensive, especially when the distance function is non-trivial to compute; for each new image classified, one has to compute the tangent distance to each of the training images, and then classify as the class of the closest. Our goal in this paper is to reduce the training set dramatically to a small set of prototype models; classification is then performed by finding the closest prototype.

## 3   PROTOTYPE MODELS

In this section we explore some ideas for generalizing the concept of a mean or centroid for a set of images, taking into account the tangent families. Such a centroid model can be used on its own, or else as a building block in a K-means or LVQ algorithm at a higher level. We will interchangeably refer to the images as points (in 256 space).

The centroid of a set of $N$ points in $d$ dimensions minimizes the average squared norm from the points:

$$M = \frac{1}{N} \sum_{i=1}^{N} X_i = \arg \min_M \sum_{i=1}^{N} \|X_i - M\|^2 \qquad (1)$$

### 3.1   TANGENT CENTROID

One could generalize this definition and ask for the point $M$ that minimizes the average squared *tangent distance*:

$$M_T = \arg \min_M \sum_{i=1}^{N} D^T(X_i, M)^2 \qquad (2)$$

This appears to be a difficult optimization problem, since computation of tangent distance requires not only the image $M$ but also its tangent basis $T_M$. Thus the criterion to be minimized is

$$C(M) = \sum_{i=1}^{N} \min_{\gamma_i, \theta_i} \|M + T(M)\gamma_i - X_i - T_i\theta_i\|^2$$

where $T(M)$ produces the tangent basis from $M$. All but the location tangent vectors are nonlinear functionals of $M$, and even without this nonlinearity, the problem to be solved is a difficult inverse functional. Fortunately a simple iterative procedure is available where we iteratively average the closest points (in tangent distance) to the current guess.

---

**Tangent Centroid Algorithm**

**Initialize:** Set $M = \frac{1}{N}\sum_{i=1}^{N} X_i$, let $T_M = T(M)$ be the derived set of tangent vectors, and $D = \sum_i D^T(X_i, M)$. Denote the current tangent centroid (tangent family) by $M(\gamma) = M + T_M\gamma$.

**Iterate:** 1. For each $i$ find a $\hat{\gamma}_i$ and $\hat{\theta}_i$ that solves $\|M + T_M\gamma - X_i(\theta)\| = \min_{\gamma,\theta}$

  2. Set $M \leftarrow \frac{1}{N}\sum_{i=1}^{N}(X_i(\hat{\theta}_i) - T_M\hat{\gamma}_i)$ and compute the new tangent subspace $T_M = T(M)$.

  3. Compute $D = \sum_i D^T(X_i, M)$

**Until:** $D$ converges.

---

Note that the first step in **Iterate** is available from the computations in the third step. The algorithm divides the parameters into two sets: $M$ in the one, and then $T_M$, $\gamma_i$ and $\theta_i$ for each $i$ in the other. It alternates between the two sets, although the computation of $T_M$ given $M$ is not the solution of an optimization problem. It seems very hard to say anything precise about the convergence or behavior of this algorithm, since the tangent vectors depend on each iterate in a nonlinear way. Our experience has always been that it converges fairly rapidly ($< 6$ iterations). A potential drawback of this algorithm is that the $T_M$ are not learned, but are implicit in $M$.

### 3.2 TANGENT SUBSPACE

Rather than define the model as a point and have it generate its own tangent subspace, we can include the subspace as part of the parametrization: $M(\gamma) = M + V\gamma$. Then we define this *tangent subspace model* as the minimizer of

$$MS(M, V) = \sum_{i=1}^{N} \min_{\gamma_i, \theta_i} \|M + V\gamma_i - X_i(\theta_i)\|^2 \tag{3}$$

over $M$ and $V$. Note that $V$ can have an arbitrary number $0 \le r \le 256$ of columns, although it does not make sense for $r$ to be too large. An iterative algorithm similar to the tangent centroid algorithm is available, which hinges on the SVD decomposition for fitting affine subspaces to a set of points. We briefly review the SVD in this context.

Let $\mathcal{X}$ be the $N \times 256$ matrix with rows the vectors $X_i - \bar{X}$ where $\bar{X} = \frac{1}{N}\sum_{i=1}^{N} X_i$. Then $SVD(\mathcal{X}) = UDV^T$ is a unique decomposition with $U_{N \times R}$ and $V_{256 \times R}$ the

orthonormal *left* and *right* matrices of *singular vectors*, and $R = \text{rank}(\mathcal{X})$. $D_{R \times R}$ is a diagonal matrix of decreasing positive *singular* values. A pertinent property of the SVD is:

> Consider finding the closest affine, rank-$r$ subspace to a set of points, or
>
> $$\min_{M, V^{(r)}, \{\theta_i\}} \sum_{i=1}^{N} \left\| X_i - M - V^{(r)} \theta_i \right\|^2$$
>
> where $V^{(r)}$ is $256 \times r$ orthonormal. The solution is given by the SVD above, with $M = \bar{X}$ and $V^{(r)}$ the first $r$ columns of $V$, and the total squared distance $\sum_{j=1}^{r} D_{jj}^2$.

The $V^{(r)}$ are also the largest $r$ principal components or eigenvectors of the covariance matrix of the $X_i$. They give in sequence directions of maximum spread, and for a given digit class can be thought of as class specific invariances.

We now present our *Tangent subspace algorithm* for solving (3); for convenience we assume $V$ is rank $r$ for some chosen $r$, and drop the superscript.

---

**Tangent subspace algorithm**

**Initialize:** Set $M = \frac{1}{N} \sum_{i=1}^{N} X_i$ and let $V$ correspond to the first $r$ right singular vectors of $\mathcal{X}$. Set $D = \sum_{j=1}^{r} D_{jj}^2$, and let the current tangent subspace model be $M(\gamma) = M + V\gamma$.

**Iterate:**  1. For each $i$ find that $\hat{\theta}_i$ which solves $\|M(\gamma) - X_i(\theta)\| = \min$

         2. Set $M \leftarrow \frac{1}{N} \sum_{i=1}^{N} (X_i(\hat{\theta}_i))$ and replace the rows of $\mathcal{X}$ by $X_i(\hat{\theta}_i) - M$. Compute the SVD of $\mathcal{X}$, and replace $V$ by the first $r$ right singular vectors.

         3. Compute $D = \sum_{j=1}^{r} D_{jj}^2$

**Until:** $D$ converges.

---

The algorithm alternates between i) finding the closest point in the tangent subspace for each image to the current tangent subspace model, and ii) computing the SVD for these closest points. Each step of the alternation decreases the criterion, which is positive and hence converges to a stationary point of the criterion. In all our examples we found that 12 complete iterations were sufficient to achieve a relative convergence ratio of 0.001.

One advantage of this approach is that we need not restrict ourselves to a seven-dimensional $V$ — indeed, we have found 12 dimensions has produced the best results. The basis vectors found for each class are interesting to view as images. Figure 1 shows some examples of the basis vectors found, and what kinds of invariances in the images they account for. These are digit specific features; for example, a prominent basis vector for the family of 2s accounts for big versus small loops.

Each of the examples shown accounts for a similar digit specific invariance. None of these changes are accounted for by the 7-dimensional tangent models, which were chosen to be digit nonspecific.

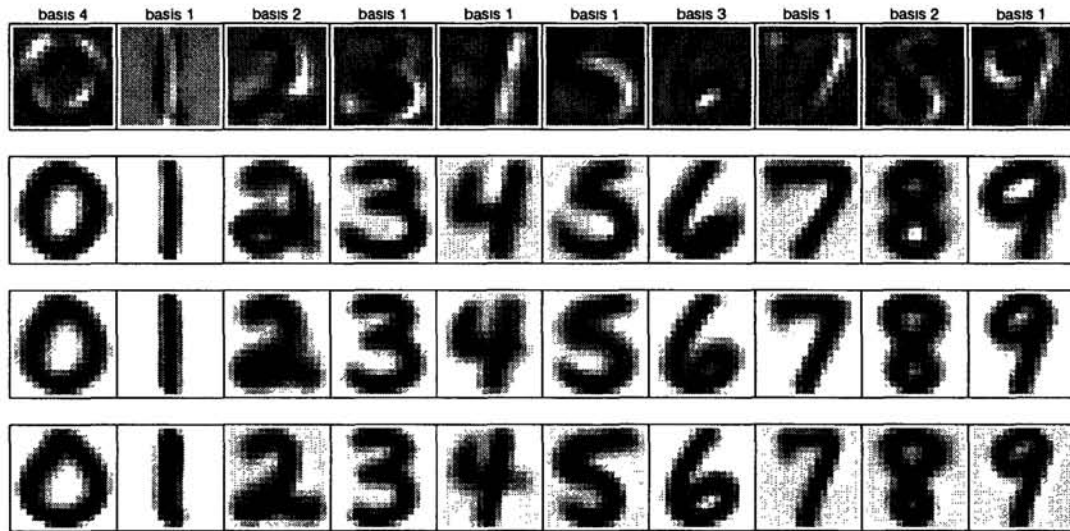

Figure 1: *Each column corresponds to a particular tangent subspace basis vector for the given digit. The top image is the basis vector itself, and the remaining 3 images correspond to the 0.1, 0.5 and 0.9 quantiles for the projection indices for the training data for that basis vector, showing a range of image models for that basis, keeping all the others at 0.*

## 4   SUBSPACE MODELS AND K-MEANS CLUSTERING

A natural and obvious extension of these single prototype-per-class models, is to use them as centroid modules in a K-means algorithm. The extension is obvious, and space permits only a rough description. Given an initial partition of the images in a class into $K$ sets:

1. Fit a separate prototype model to each of the subsets;
2. Redefine the partition based on closest tangent distance to the prototypes found in step 1.

In a similar way the tangent centroid or subspace models can be used to seed LVQ algorithms (Kohonen 1989), but so far we have not much experience with them.

## 5   RESULTS

Table 1 summarizes the results for some of these models. The first two lines correspond to a SVD model for the images fit by ordinary least squares rather than least tangent squares. The first line classifies using Euclidean distance to this model, the second using tangent distance. Line 3 fits a single 12-dimensional *tangent subspace* model per class, while lines 4 and 5 use 12-dimensional tangent subspaces as cluster

Table 1: *Test errors for a variety of situations. In all cases the training data were 7291 USPS handwritten digits, and the test data the "official" 2007 USPS test digits. Each entry describes the model used in each class, so for example in row 5 there are 5 models per class, hence 50 in all.*

|   | Prototype | Metric | # Prototypes/Class | Error Rate |
|---|---|---|---|---|
| 0 | 1-NN | Euclidean | ≈ 700 | 0.053 |
| 1 | 12 dim SVD subspace | Euclidean | 1 | 0.055 |
| 2 | 12 dim SVD subspace | Tangent | 1 | 0.045 |
| 3 | 12 dim Tangent subspace | Tangent | 1 | 0.041 |
| 4 | 12 dim Tangent subspace | Tangent | 3 | 0.038 |
| 5 | 12 dim Tangent subspace | Tangent | 5 | 0.038 |
| 6 | Tangent centroid | Tangent | 20 | 0.038 |
| 7 | (4) ∪ (6) | Tangent | 23 | 0.034 |
| 8 | 1-NN | Tangent | ≈ 700 | 0.026 |

centers within each class. We tried other dimensions in a variety of settings, but 12 seemed to be generally the best. Line 6 corresponds to the *tangent centroid* model used as the centroid in a 20-means cluster model per class; the performance compares with with K=3 for the subspace model. Line 7 combines 4 and 6, and reduces the error even further. These limited experiments suggest that the tangent subspace model is preferable, since it is more compact and the algorithm for fitting it is on firmer theoretical grounds.

Figure 4 shows some of the misclassified examples in the test set. Despite all the matching, it seems that Euclidean distance still fails us in the end in some of these cases.

# 6  DISCUSSION

Gold, Mjolsness & Rangarajan (1994) independently had the idea of using "domain specific" distance measures to seed K-means clustering algorithms. Their setting was slightly different from ours, and they did not use subspace models. The idea of classifying points to the closest subspace is found in the work of Oja (1989), but of course not in the context of tangent distance.

We are using Euclidean distance in conjunction with tangent distance. Since neighboring pixels are correlated, one might expect that a metric that accounted for the correlation might do better. We tried several variants using Mahalanobis metrics in different ways, but with no success. We also tried to incorporate information about where the images project in the tangent subspace models into the classification rule. We thus computed two distances: 1) tangent distance to the subspace, and 2) Mahalanobis distance within the subspace to the centroid for the subspace. Again the best performance was attained by ignoring the latter distance.

In conclusion, learning tangent centroid and subspace models is an effective way

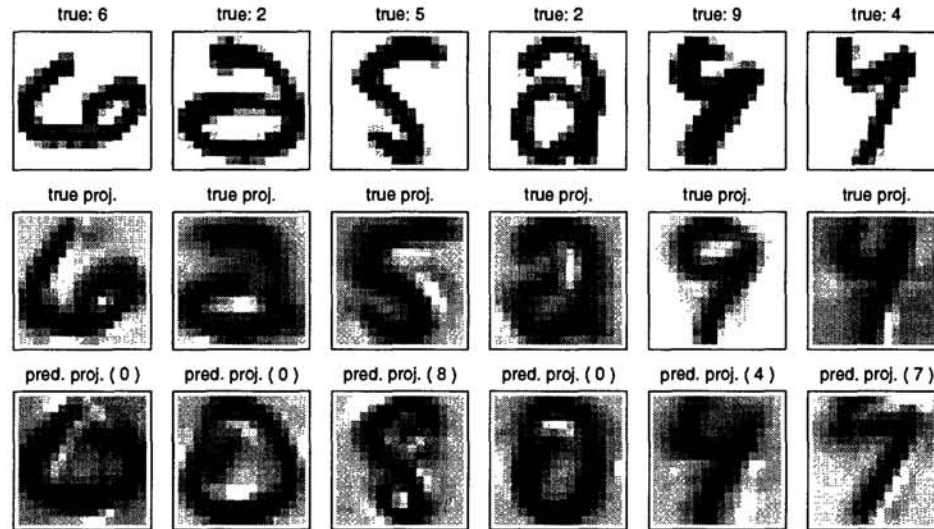

Figure 2: *Some of the errors for the test set corresponding to line (3) of table 4. Each case is displayed as a column of three images. The top is the true image, the middle the tangent projection of the true image onto the subspace model of its class, the bottom image the tangent projection of the image onto the winning class. The models are sufficiently rich to allow distortions that can fool Euclidean distance.*

to reduce the number of prototypes (and thus the cost in speed and memory) at a slight expense in the performance. In the extreme case, as little as one 12 dimensional tangent subspace per class and the tangent distance is enough to outperform classification using $\approx$ 700 prototypes per class and the Euclidean distance (4.1% versus 5.3% on the test data).

## Footnotes

*This work was performed while Trevor Hastie was a member of the Statistics and Data Analysis Research Group, AT&T Bell Laboratories, Murray Hill, NJ 07974.

# References

Gold, S., Mjolsness, E. & Rangarajan, A. (1994), Clustering with a domain specific distance measure, *in* 'Advances in Neural Information Processing Systems', Morgan Kaufman, San Mateo, CA.

Kohonen, T. (1989), *Self-Organization and Associative Memory (3rd edition)*, Springer-Verlag, Berlin.

Oja, E. (1989), 'Neural networks, principal components, and subspaces', *International Journal Of Neural Systems* 1(1), 61–68.

Säckinger, E. (1992), Recurrent networks for elastic matching in pattern recognition, Technical report, AT&T Bell Laboratories.

Simard, P. Y., LeCun, Y. & Denker, J. (1993), Efficient pattern recognition using a new transformation distance, *in* 'Advances in Neural Information Processing Systems', Morgan Kaufman, San Mateo, CA, pp. 50–58.